# Priors for Diversity in Generative Latent Variable Models

**James Y. Zou**
School of Engineering and Applied Sciences
Harvard University
Cambridge, MA 02138
`jzou@fas.harvard.edu`

**Ryan P. Adams**
School of Engineering and Applied Sciences
Harvard University
Cambridge, MA 02138
`rpa@seas.harvard.edu`

## Abstract

Probabilistic latent variable models are one of the cornerstones of machine learning. They offer a convenient and coherent way to specify prior distributions over unobserved structure in data, so that these unknown properties can be inferred via posterior inference. Such models are useful for exploratory analysis and visualization, for building density models of data, and for providing features that can be used for later discriminative tasks. A significant limitation of these models, however, is that draws from the prior are often highly redundant due to i.i.d. assumptions on internal parameters. For example, there is no preference in the prior of a mixture model to make components non-overlapping, or in topic model to ensure that co-occurring words only appear in a small number of topics. In this work, we revisit these independence assumptions for probabilistic latent variable models, replacing the underlying i.i.d. prior with a determinantal point process (DPP). The DPP allows us to specify a preference for diversity in our latent variables using a positive definite kernel function. Using a kernel between probability distributions, we are able to define a DPP on probability measures. We show how to perform MAP inference with DPP priors in latent Dirichlet allocation and in mixture models, leading to better intuition for the latent variable representation and quantitatively improved unsupervised feature extraction, without compromising the generative aspects of the model.

## 1   Introduction

The probabilistic generative model is an important tool for statistical learning because it enables rich data to be explained in terms of simpler latent structure. The discovered structure can be useful in its own right, for explanatory purposes and visualization, or it may be useful for improving generalization to unseen data. In the latter case, we might think of the inferred latent structure as providing a feature representation that summarizes complex high-dimensional interaction into a simpler form.

The core assumption behind the use of latent variables as features, however, is that the salient statistical properties discovered by unsupervised learning will be useful for discriminative tasks. This requires that the features span the space of possible data and represent diverse characteristics that may be important for discrimination. Diversity, however, is difficult to express within the generative framework. Most often, one builds a model where the feature representations are independent *a priori*, with the hope that a good fit to the data will require employing a variety of latent variables.

There is reason to think that this does not always happen in practice, and that during unsupervised learning, model capacity is often spent improving the density around the common cases, not allocating new features. For example, in a generative clustering model based on a mixture distribution, multiple mixture components will often be used for a single "intuitive group" in the data, simply because the shape of the component's density is not a close fit to the group's distribution. A generative

mixture model will happily use many of its components to closely fit the density of a single group of data, leading to a highly redundant feature representation. Similarly, when applied to a text corpus, a topic model such as latent Dirichlet allocation [1] will place large probability mass on the same stop words across many topics, in order to fine-tune the probability assigned to the common case. In both of these situations, we would like the latent groupings to uniquely correspond to characteristics of the data: that a group of data should be explained by one mixture component, and that common stop words should be one category of words among many. This intuition expresses a need for diversity in the latent parameters of the model that goes beyond what is highly likely under the posterior distribution implied by an independent prior.

In this paper, we propose a modular approach to diversity in generative probabilistic models by replacing the independent prior on latent parameters with a *determinantal point process* (DPP). The determinantal point process enables a modeler to specify a notion of similarity on the space of interest, which in this case is a space of possible latent distributions, via a positive definite kernel. The DPP then assigns probabilities to particular configurations of these distributions according to the determinant of the Gram matrix. This construction naturally leads to a generative latent variable model in which diverse sets of latent parameters are preferred over redundant sets.

The determinantal point process is a convenient statistical tool for constructing a tractable point process with repulsive interaction. The DPP is more general than the Poisson process (see, e.g., [2]), which has no interaction, but more tractable than Strauss [3] and Gibbs/Markov [4] processes (at the cost of only being able to capture anticorrelation). Hough *et al.* [5] provides a useful survey of probabilistic properties of the determinantal point process, and for statistical properties, see, e.g., Scardicchio *et al.* [6] and Lavancier *et al.* [7]. There has also been recent interest in using the DPP within machine learning for modeling sets of structures [8], and for conditionally producing diverse collections of objects [9]. The approach we propose here is different from this previous work in that we are suggesting the use of a determinantal point process within a hierarchical model, and using it to enforce diversity among latent variables, rather than as a mechanism for diversity across directly observed discrete structures.

## 2   Diversity in Generative Latent Variable Models

In this paper we consider generic directed probabilistic latent variable models that produce distributions over a set of $N$ data, denoted $\{x_n\}_{n=1}^{N}$, which live in a sample space $\mathcal{X}$. Each of these data has a latent discrete label $z_n$, which takes a value in $\{1, 2, \cdots, J\}$. The latent label indexes into a set of parameters $\{\theta_j\}_{j=1}^{J}$. The parameters determined by $z_n$ then produce the data according to a distribution $f(x_n \mid \theta_{z_n})$. Typically we use independent priors for the $\theta_j$, here denoted by $\pi(\cdot)$, but the distribution over the latent indices $z_n$ may be more structured. Taken together this leads to the generic joint distribution:

$$p(\{x_n, z_n\}_{n=1}^{N}, \{\theta_j\}_{j=1}^{J}) = p(\{z_n\}_{n=1}^{N}) \left[ \prod_{n=1}^{N} f(x_n \mid \theta_{z_n}) \right] \prod_{j=1}^{J} \pi(\theta_j). \tag{1}$$

The details of each distribution are problem-specific, but this general framework appears in many contexts. For example, in a typical mixture model, the $z_n$ are drawn independently from a multinomial distribution and the $\theta_j$ are the component-specific parameters. In an admixture model such as latent Dirichlet allocation (LDA) [1], the $\theta_j$ may be "topics", or distributions over words. In an admixture, the $z_n$ may share structure based on, e.g., being words within a common set of documents.

These models are often thought of as providing a principled approach for feature extraction. At training time, one either finds the maximum of the posterior distribution $p(\{\theta_j\}_{j=1}^{J} \mid \{x_n\}_{n=1}^{N})$ or collects samples from it, while integrating out the data-specific latent variables $z_n$. Then when presented with a test case $x^\star$, one can construct a conditional distribution over the corresponding unknown variable $z^\star$, which is now a "feature" that might usefully summarize many related aspects of $x^\star$. However, this interpretation of the model is suspect; we have not asked the model to make the $z_n$ variables explanatory, except as a byproduct of improving the training likelihood. Different $\theta_j$ may assign essentially identical probabilities to the same datum, resulting in ambiguous features.

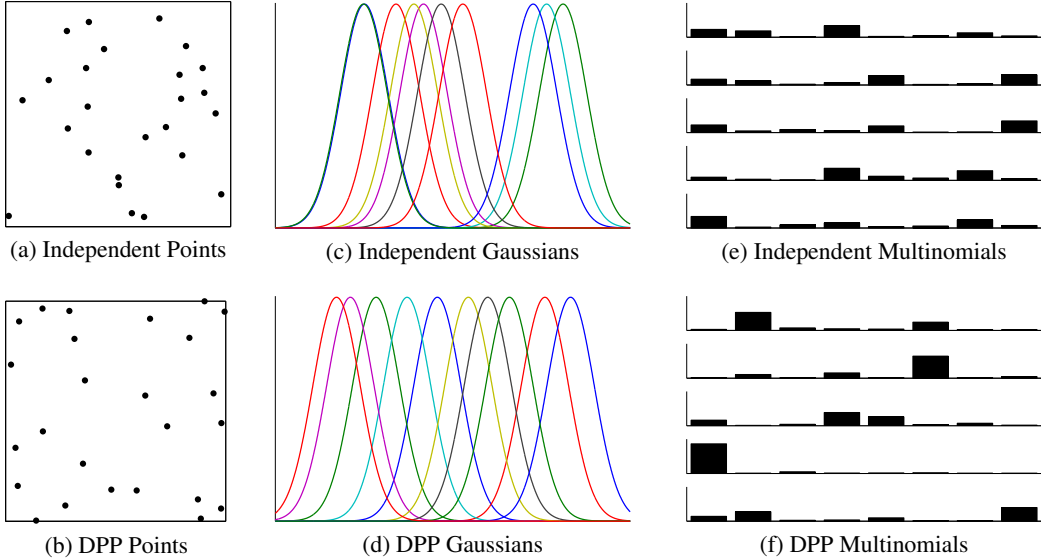

|                          |                            |                               |
|:------------------------:|:--------------------------:|:-----------------------------:|
| (a) Independent Points   | (c) Independent Gaussians  | (e) Independent Multinomials  |
| (b) DPP Points           | (d) DPP Gaussians          | (f) DPP Multinomials          |

Figure 1: Illustrations of the determinantal point process prior. (a) 25 independent uniform draw in the unit square; (b) a draw from a DPP with 25 points; (c) ten Gaussian distributions with means uniformly drawn on the unit interval; (d) ten Gaussian distributions with means distributed according to a DPP using the probability product kernel; (e) five random discrete distributions; (f) five random discrete distributions drawn from a DPP on the simplex with the probability product kernel [10].

## 2.1 Measure-Valued Determinantal Point Process

In this work we propose an alternative to the independence assumption of the standard latent variable model. Rather than specifying $p(\{\theta_j\}_{j=1}^J) = \prod_j \pi(\theta_j)$, we will construct a determinantal point process on sets of component-specific distributions $\{f(x \mid \theta_j)\}_{j=1}^J$. Via the DPP, it will be possible for us to specify a preference for sets of distributions that have minimal overlap, as determined via a positive-definite kernel function between distributions. In the case of the simple parametric families for $f(\cdot)$ that we consider here, it is appropriate to think of the DPP as providing a "diverse" set of parameters $\boldsymbol{\theta} = \{\theta_j\}_{j=1}^J$, where the notion of diversity is expressed entirely in terms of the resulting probability measure on the sample space $\mathcal{X}$. After MAP inference with this additional structure, the hope is that the $\theta_j$ will explain substantially different regions of $\mathcal{X}$ — appropriately modulated by the likelihood — and lead to improved, non-redundant feature extraction at test time.

We will use $\Theta$ to denote the space of possible $\theta$. A realization from a *point process* on $\Theta$ produces a random finite subset of $\Theta$. To construct a determinantal point process, we first define a positive definite kernel on $\Theta$, which we denote $K : \Theta \times \Theta \to \mathbb{R}$. The probability density associated with a particular finite $\boldsymbol{\theta} \subset \Theta$ is given by

$$p(\boldsymbol{\theta} \subset \Theta) \propto |\mathbf{K}_{\boldsymbol{\theta}}|, \tag{2}$$

where $\mathbf{K}_{\boldsymbol{\theta}}$ is the $|\boldsymbol{\theta}| \times |\boldsymbol{\theta}|$ positive definite Gram matrix that results from applying $K(\theta, \theta')$ to the elements of $\boldsymbol{\theta}$. The eigenspectrum of the kernel on $\Theta$ must be bounded to $[0, 1]$. The kernels we will focus on in this paper are composed of two parts: 1) a positive definite *correlation function* $R(\theta, \theta')$, where $R(\theta, \theta) = 1$, and 2) the "prior kernel" $\sqrt{\pi(\theta)\pi(\theta')}$, which expresses our marginal preferences for some parameters over others. These are combined to form the kernel of interest:

$$K(\theta, \theta') = R(\theta, \theta') \sqrt{\pi(\theta)\pi(\theta')}, \tag{3}$$

which leads to the matrix form $\mathbf{K}_{\boldsymbol{\theta}} = \mathbf{\Pi} \mathbf{R}_{\boldsymbol{\theta}} \mathbf{\Pi}$, where $\mathbf{\Pi} = \text{diag}([\sqrt{\pi(\theta_1)}, \sqrt{\pi(\theta_2)}, \cdots])$.

Note that if $R(\theta, \theta') = 0$ when $\theta \neq \theta'$, this construction recovers the Poisson process with intensity measure $\pi(\theta)$. Note also in this case that if the cardinality of $\boldsymbol{\theta}$ is predetermined, then this recovers the traditional independent prior. More interesting, however, are $R(\theta, \theta')$ with off-diagonal structure that induces interaction within the set. Such kernels will always induce repulsion of the points so that diverse subsets of $\Theta$ will tend to have higher probability under the prior. See Fig. 1 for illustrations of the difference between independent samples and the DPP for several different settings.

## 2.2 Kernels for Probability Distributions

The determinantal point process framework allows us to construct a generative model for repulsion, but as with other kernel-based priors, we must define what "repulsion" means. A variety of positive definite functions on probability measures have been defined, but in this work we will use the *probability product* kernel [10]. This kernel is a natural generalization of the inner product for probability distributions. The basic kernel has the form

$$K(\theta, \theta'\,;\,\rho) = \int_{\mathcal{X}} f(x\,|\,\theta)^\rho f(x\,|\,\theta')^\rho \,\mathrm{d}x \tag{4}$$

for $\rho > 0$. As we require a correlation kernel, we use the normalized variant given by

$$R(\theta, \theta'\,;\,\rho) = K(\theta, \theta'\,;\,\rho)/\sqrt{K(\theta, \theta\,;\,\rho)K(\theta', \theta'\,;\,\rho)}. \tag{5}$$

This kernel has convenient closed forms for several distributions of interest, which makes it an ideal building block for the present model.

## 2.3 Replicated Determinantal Point Process

A property that we often desire from our prior distributions is that they have the ability to become arbitrarily strong. That is, under the interpretation of a Bayesian prior as "inferences from previously-seen data", we would like to be able to imagine an arbitrary amount of such data and construct a highly-informative prior when appropriate. Unfortunately, the standard determinantal point process does not provide a knob to turn to increase its strength arbitrarily.

For example, take a DPP on a Euclidean space and consider a point $t$, an arbitrary unit vector $w$ and a small scalar $\epsilon$. Construct two pairs of points using a $\delta > 1$: a "near" pair $\{t, t + \epsilon w\}$, and a "far" pair $\{t, t + \epsilon\delta w\}$. We wish to find some small $\epsilon$ such that the "far" configuration is arbitrarily more likely than the "near" configuration under the DPP. That is, we would like the ratio of determinants

$$r(\epsilon) = \frac{p(\{t, t + \epsilon\delta w\})}{p(\{t, t + \epsilon w)\})} = \frac{1 - R(t, t + \epsilon\delta w)^2}{1 - R(t, t + \epsilon w))^2}, \tag{6}$$

to be unbounded as $\epsilon$ approaches zero. The objective is to have a scaling parameter that can cause the determinantal prior to be arbitrarily strong relative to the likelihood terms. If we perform a Taylor expansion of the numerator and denominator around $\epsilon = 0$, we get

$$r(\epsilon) \approx \frac{1 - (R(t, t) + 2\delta w\epsilon \left[\frac{\mathrm{d}}{\mathrm{d}\tilde{t}}R(t, \tilde{t})\right]_{\tilde{t}=t})}{1 - (R(t, t) + 2w\epsilon \left[\frac{\mathrm{d}}{\mathrm{d}\tilde{t}}R(t, \tilde{t})\right]_{\tilde{t}=t})} = \delta. \tag{7}$$

We can see that, when near zero, this ratio captures the difference in distances, but not in a way that can be rescaled to greater effect. This means that there exist finite data sets that we cannot overwhelm with any DPP prior. To address this issue, we augment the determinantal point process with an additional parameter $\lambda > 0$, so that the probability of a finite subset $\boldsymbol{\theta} \subset \Theta$ becomes

$$p(\boldsymbol{\theta} \subset \Theta) \propto |\mathbf{K}_{\boldsymbol{\theta}}|^\lambda. \tag{8}$$

For integer $\lambda$, it can be viewed as a set of $\lambda$ identical "replicated realizations" from determinantal point processes, leaving our generative view intact. The replicate of $\boldsymbol{\theta}$ is just $\boldsymbol{\theta}_\lambda = \{\lambda \text{ copies of } \boldsymbol{\theta}\}$ and the corresponding $\mathbf{K}_{\boldsymbol{\theta}_\lambda}$ is a $\lambda|\boldsymbol{\theta}| \times \lambda|\boldsymbol{\theta}|$ block diagonal matrix where each block is a replicate of $\mathbf{K}_{\boldsymbol{\theta}}$. This maps well onto the view of a prior as pseudo-data; our replicated DPP asserts that we have seen $\lambda$ previous such data sets. As in other pseudo-count priors, we do not require in practice that $\lambda$ be an integer, and under a penalized log likelihood view of MAP inference, it can be interpreted as a parameter for increasing the effect of the determinantal penalty.

## 2.4 Determinantal Point Process as Regularization.

In addition to acting as a prior over distributions in the generative setting, we can also view the DPP as a new type of "diversity" regularizer on learning. The goal is to solve

$$\boldsymbol{\theta}^\star = \underset{\boldsymbol{\theta} \subset \Theta}{\operatorname{argmin}}\, \mathcal{L}(\boldsymbol{\theta}; \{x_n\}_{n=1}^N) - \lambda \ln |\mathbf{K}_{\boldsymbol{\theta}}|, \tag{9}$$

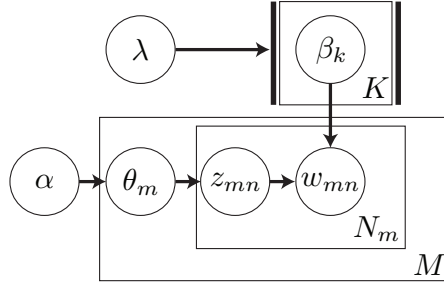

Figure 2: Schematic of DPP-LDA. We replace the standard plate notation for i.i.d topics in LDA with a "double-struck plate" to indicate a determinantal point process.

choosing the best *set* of parameters $\boldsymbol{\theta}$ from $\Theta$. Here $\mathcal{L}(\cdot)$ is a loss function that depends on the data and the discrimination function, with parameters $\boldsymbol{\theta}$. From Eqn. (3),

$$\ln |\mathbf{K}_{\boldsymbol{\theta}}| = \ln |\mathbf{R}_{\boldsymbol{\theta}}| + \sum_{\theta_j \in \boldsymbol{\theta}} \ln \pi(\theta_j). \tag{10}$$

If $\mathcal{L}(\cdot) = -\ln p(\{x_n\}_{n=1}^N | \boldsymbol{\theta})$, then the resulting optimization is simply MAP estimation. In this framework, we can combine the DPP penalty with any other regularizer on $\theta$, for example the sparsity-inducing $\ell_1$ regularizer. In the following sections, we give empirical evidence that this diversity improves generalization performance.

## 3 MAP Inference

In what follows, we fix the cardinality $|\boldsymbol{\theta}|$. Viewing the kernel $\mathbf{K}_{\boldsymbol{\theta}}$ as a function of $\boldsymbol{\theta}$, the gradient $\frac{\partial}{\partial \theta} \log |\mathbf{K}_{\boldsymbol{\theta}}| = \text{trace}(\mathbf{K}_{\boldsymbol{\theta}}^{-1} \frac{\partial \mathbf{K}_{\boldsymbol{\theta}}}{\partial \theta})$. This allows application of general gradient-based optimization algorithms for inference. In particular, we can optimize $\boldsymbol{\theta}$ as a modular component within an off-the-shelf expectation maximization (EM) algorithm. Here we examine two illustrative examples of generative latent variable models into which we can directly plug our DPP-based prior.

**Diversified Latent Dirichlet Allocation** Latent Dirichlet allocation (LDA) [1] is an immensely popular admixture model for text and, increasingly, for other kinds of data that can be treated as a "bag of words". LDA constructs a set of topics — distributions over the vocabulary — and asserts that each word in the corpus is explained by one of these topics. The topic-word assignments are unobserved, but LDA attempts to find structure by requiring that only a small number of topics be represented in any given document.

In the standard LDA formulation, the topics are $K$ discrete distributions $\beta_k$ over a vocabulary of size $V$, where $\beta_{kv}$ is the probability of topic $k$ generating word $v$. There are $M$ documents and the $m$th document has $N_m$ words. Document $m$ has a latent multinomial distribution over topics, denoted $\theta_m$ and each word in the document $w_{mn}$ has a topic index $z_{mn}$ drawn from $\theta_m$. While classical LDA uses independent Dirichlet priors for the $\beta_k$, here we "diversify" latent Dirichlet allocation by replacing this prior with a DPP. That is, we introduce a correlation kernel

$$R(\beta_k, \beta_{k'}) = \frac{\sum_{v=1}^V (\beta_{kv} \, \beta_{k'v})^\rho}{\sqrt{\sum_{v=1}^V \beta_{kv}^{2\rho}} \sqrt{\sum_{v=1}^V \beta_{k'v}^{2\rho}}}, \tag{11}$$

which approaches one as $\beta_k$ becomes more similar to $\beta_{k'}$. In the application below of DPP-LDA, we use $\rho = 0.5$. We use $\pi(\beta_k) = \text{Dirichlet}(\alpha)$, and write the resulting prior as $p(\boldsymbol{\beta}) \propto |\mathbf{K}_{\boldsymbol{\beta}}|$. We call this model "DPP-LDA", and illustrate it with a graphical model in Figure 2. We use a "double-struck plate" in the graphical model to represent the DPP, and highlight how it can be used as a drop-in replacement for the i.i.d. assumption.

To perform MAP learning of this model, we construct a modified version of the standard variational EM algorithm. As in variational EM for LDA, we define a factored approximation

$$q(\theta_m, z_m | \gamma_m, \phi_m) = q(\theta_m | \gamma_m) \prod_{n=1}^N q(z_{mn} | \phi_{mn}). \tag{12}$$

| LDA | | DPP-LDA | | | | | |
|---|---|---|---|---|---|---|---|
| typical | | "stop words" | | "Christianity" | "space" | "OS" | "politics" |
| the | the | the | and | jesus | space | file | ms |
| to | to | of | in | matthew | nasa | pub | myers |
| and | and | that | at | prophecy | astronaut | usr | god |
| in | it | you | from | christians | mm | available | president |
| of | of | by | some | church | mission | export | but |
| is | is | one | their | messiah | pilot | font | package |
| it | in | all | with | psalm | shuttle | lib | options |
| for | that | but | your | isaiah | military | directory | dee |
| that | for | do | who | prophet | candidates | format | believe |
| can | you | my | which | lord | ww | server | groups |

Table 1: Top ten words from representative topics learned in LDA and DPP-LDA.

In this approximation, each document $m$ has a Dirichlet approximation to its posterior over topics, given by $\gamma_m$. $\phi_m$ is an $N \times K$ matrix in which the $n$th row, denoted $\phi_{mn}$, is a multinomial distribution over topics for word $w_{mn}$. For the current estimate of $\beta_{kv}$, $\gamma_m$ and $\phi_m$ are iteratively optimized. See Blei *et al.* [1] for more details. Our extension of variational EM to include the DPP does not require alteration of these steps.

The inclusion of the determinantal point process prior does, however, effect the maximization step. The diversity prior introduces an additional penalty on $\boldsymbol{\beta}$, so that the M-step requires solving

$$\boldsymbol{\beta}^\star = \underset{\boldsymbol{\beta}}{\mathrm{argmax}} \left\{ \sum_{m=1}^{M} \sum_{n=1}^{N_m} \sum_{k=1}^{K} \sum_{v=1}^{V} \phi_{mnk} w_{mn}^{(v)} \ln \beta_{kv} + \lambda \ln |\mathbf{K}_{\boldsymbol{\beta}}| \right\}, \qquad (13)$$

subject to the constraints that each row of $\boldsymbol{\beta}$ sum to 1. For $\lambda = 0$, this optimization procedure yields the standard update for vanilla LDA, $\beta_{kv}^\star \propto \sum_{m=1}^{M} \sum_{n=1}^{N_m} \phi_{mnk} w_{mn}^{(v)}$. For $\lambda > 0$ we use gradient descent to find a local optimal $\boldsymbol{\beta}$.

**Diversified Gaussian Mixture Model** The mixture model is a popular model for generative clustering and density estimation. Given $J$ components, the probability of the data is given by

$$p(x_n \mid \boldsymbol{\theta}) = \sum_{j=1}^{J} \chi_j \, f(x_n \mid \theta_j). \qquad (14)$$

Typically, the $\theta_k$ are taken to be independent in the prior. Here we examine determinantal point process priors for the $\theta_k$ in the case where the components are Gaussian.

For Gaussian mixture models, the DPP prior is particularly tractable. As in DPP-LDA, we use the probability product kernel, which in this case also has a convenient closed form [10]. Let $f_1 = \mathcal{N}(\mu_1, \Sigma_1)$ and $f_2 = \mathcal{N}(\mu_2, \Sigma_2)$ be two Gaussians, the product kernel is:

$$K(f_1, f_2) = (2\pi)^{(1-2\rho)\frac{D}{2}} \rho^{-\frac{D}{2}} |\hat{\Sigma}|^{\frac{1}{2}} (|\Sigma_1||\Sigma_2|)^{-\frac{\rho}{2}} \exp(-\frac{\rho}{2}(\mu_1^T \Sigma_1^{-1} \mu_1 + \mu_2^T \Sigma_2^{-1} \mu_2 - \hat{\mu}^T \hat{\Sigma} \hat{\mu}))$$

where $\hat{\Sigma} = (\Sigma_1 + \Sigma_2)^{-1}$ and $\hat{\mu} = \Sigma_1^{-1} \mu_1 + \Sigma_2^{-1} \mu_2$. In the special case of a fixed, isotropic covariance $\sigma^2 I$ and $\rho = 1$, the kernel is

$$K(f(\cdot \mid \mu), f(\cdot \mid \mu')) = \frac{1}{(4\pi\sigma^2)^{D/2}} e^{-||\mu - \mu'||^2/(4\sigma^2)} \qquad (15)$$

where $D$ is the data dimensionality.

In the standard EM algorithm for Gaussian mixtures, one typically introduces latent binary variables $z_{nj}$, which indicate that datum $n$ belongs to component $j$. The E-step computes the responsibility vector $\gamma(z_{nj}) = E[z_{nj}] \propto \chi_j \mathcal{N}(x_n | \mu_j, \Sigma_j)$. This step is identical for DPP-GMM. The update for the component weights is also the same: $\chi_j = \frac{1}{N} \sum_{n=1}^{N} \gamma(z_{nj})$. The difference between this procedure and the standard EM approach is that the M-step for the DPP-GMM optimizes the objective function (summarizing $\{\mu_j, \Sigma_j\}_{j=1}^{J}$ by $\boldsymbol{\theta}$ for clarity):

$$\boldsymbol{\theta}^\star = \underset{\boldsymbol{\theta} \in \Theta}{\mathrm{argmax}} \left\{ \sum_{n=1}^{N} \sum_{j=1}^{J} \gamma(z_{nj}) \left[ \ln \chi_j + \ln \mathcal{N}(x_n | \mu_j, \Sigma_j) \right] + \lambda \ln |\mathbf{K}_{\boldsymbol{\theta}}| \right\}. \qquad (16)$$

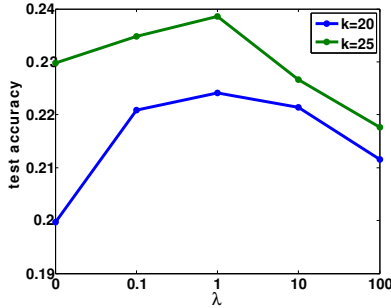
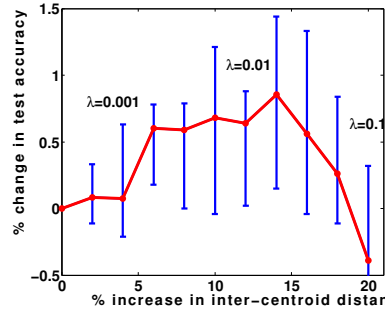

Figure 3: Effect of $\lambda$ on classification error.      Figure 4: Effect of centroid distance on test error.

Closely related to DPP-GMM is *DPP-$K$-means*. The kernel acts on the set of centroids as in Eqn. (15), with $\sigma^2$ now just a constant scaling term. Let $\boldsymbol{\theta} = \{\mu_j\}$ and $z_{nj}$ be the hard assignment indicator, the maximization step is:

$$\boldsymbol{\theta}^\star = \underset{\boldsymbol{\theta} \in \Theta}{\operatorname{argmax}} \left\{ \sum_{n=1}^{N} \sum_{j=1}^{J} z_{nj} ||x_n - \mu_j||^2 + \lambda \ln |\mathbf{K}_{\boldsymbol{\theta}}| \right\}. \tag{17}$$

With the product kernel, the similarity between two Gaussians decays exponentially as the distance between their means increases. In practice, we find that when the number of mixture components $|\boldsymbol{\theta}|$ is large, $\mathbf{K}_{\boldsymbol{\theta}}$ is well approximated by a sparse matrix.

## 4 Experiment I: diversified topic modeling.

We tested LDA and DPP-LDA on the unfiltered 20 Newsgroup corpus, without removing any stop-words. A common frustration with vanilla LDA is that applying LDA to unfiltered data returns topics that are dominated by stop-words. This frustrating phenomenon occurs even as the number of topics is varied from $K = 5$ to $K = 50$. The first two columns of Table 1 show the ten most frequent words from two representative topics learned by LDA using $K = 25$. Stop-words occur frequently across all documents and thus are unhelpfully correlated with topic-specific informative keywords. We repeated the experiments after removing a list of 669 most common stop-words. However, the topics inferred by regular LDA are still dominated by secondary stop-words that are not informative.

*DPP-LDA automatically groups common stop words into a few topics.* By finding stop-word-specific topics, the majority of the remaining topics are available for more informative words. Table 1 shows a sample of topics learned by DPP-LDA on the unfiltered 20 Newsgroup corpus ($K = 25$, $\lambda = 10^4$). As we vary $K$ or increase $\lambda$ we observe robust grouping of stop-words into a few topics. High frequency words that are common across many topics significantly increase the similarity between the topics, as measured by the product kernel on the $\boldsymbol{\beta}$ distributions. This similarity incurs a large penalty in DPP and so the objective actively pushes the parameters of LDA away from regions where stop words occupy large probability mass across many topics.

*Features learned from DPP-LDA leads to better document classification.* It is common to use the $\gamma_m$, the document specific posterior distribution over topics, as feature vectors in document classification. We inferred $\{\gamma_{m,train}\}$ on training documents from DPP-LDA variational EM, and then trained a support vector machine (SVM) classifier on $\{\gamma_{m,train}\}$ with the true topic labels from 20 Newsgroups. On test documents, we fixed the parameters $\alpha$ and $\boldsymbol{\beta}$ to the values inferred from the training set, and used variational EM to find MAP estimates of $\{\gamma_{m,test}\}$. The mean test classification accuracy for a range of $\lambda$ values is plotted in Figure 3. The setting $\lambda = 0$ corresponds to vanilla LDA. In each trial, we use the same training set for DPP-LDA on a range of $\lambda$ values. DPP-LDA with $\lambda = 1$ consistently outperforms LDA in test classification ($p < 0.001$ binomial test). Large values of $\lambda$ decrease classification performance.

## 5 Experiment II: diverse clustering.

Mixture models are often a useful way to learn features for classification. The recent work of Coates *et al.* [11], for example, shows that even simple $K$-means works well as a method of extracting

| training set size | K | $K$-means | DPP $K$-means | gain (%) | $\lambda$ |
|---|---|---|---|---|---|
| 500 | 30 | 34.81 | **36.21** | 1.4 | 0.01 |
| 1000 | 30 | 43.32 | **44.27** | 0.95 | 0.01 |
| 2000 | 60 | 52.05 | **52.55** | 0.50 | 0.01 |
| 5000 | 150 | 61.03 | **61.23** | 0.20 | 0.001 |
| 10000 | 300 | 66.36 | **66.65** | 0.29 | 0.001 |

Table 2: Test classification accuracy on CIFAR-10 dataset.

features for image labeling. In that work, $K$-means gave state of art results on the CIFAR-10 object recognition task. Coates *et al.* achieved these results using a patch-wise procedure in which random patches are sampled from images for training. Each patch is a 6-by-6 square, represented as a point in a 36 dimensional vector space. Patches from the training images are combined and clustered using $K$-means. Each patch is then represented by a binary $K$-dimensional feature vector: the $k^{th}$ entry is one if the patch is closer to the centroid $k$ than its average distance to centroids. Roughly half of the feature entries are zero. Patches from the same image are then pooled to construct one feature vector for the whole image. An SVM is trained on these image features to perform classification.

We reason that DPP-$K$-means may produce more informative features since the cluster centroids will repel each other into more distinct positions in pixel space. We replicated the experiments from Coates *et al.*, using their publicly-available code for identical pre- and post-processing. With this setup, $\lambda = 0$ recovers regular $K$-means, and reproduces the results from Coates *et al.* [11]. We applied DPP-$K$-means to the CIFAR-10 dataset, while varying the size of the training set. For each training set size, we ran regular $K$-means for a range of values of $K$ and select the $K$ that gives the best test accuracy for $K$-means. Then we compare the performance with DPP-$K$-means using the same $K$. For up to 10000 images in the training set, DPP-$K$-means leads to better test classification accuracy compared to the simple $K$-means. The comparisons are performed on matched settings: for a given randomly sampled training set and a centroid initialization, we generate the centroids from both $K$-means and DPP-$K$-means. The two sets of centroids were used to extract features and train classifiers, which are then tested on the same test set of images. DPP-$K$-means consistently outperforms $K$-means in generalization accuracy ($p < 0.001$ binomial test). For example, for training set of size 1000, with $k = 30$, we ran 100 trials, each with an random training set and initialization, DPP-$K$-means outperformed $K$-means in 94 trials. As expected given its role as a regularizer, improvement from DPP-$K$-means is more significant for smaller training sets. For the full CIFAR-10 with 50000 training images, DPP-$K$-means does not consistently outperform $K$-means.

Next we ask if there is a pattern between how far the DPP pushes apart the centroids and classification accuracy on the test set. Focusing on 1000 training images and $k = 30$, for each randomly sampled training set and centroid initialization, we compute the mean inter-centroid distance for $K$-means and DPP-$K$-means. We compute the test accuracy for each set of centroids. Fig. 4 bins the relative increase in inter-centroid distance into 10 bins. For each bin, we show the $25^{th}$, $50^{th}$, and $75^{th}$ percentile of changes in test accuracy. Test accuracy is maximized when the inter-centroid distances increase by about $14\%$ from $K$-means centroids, corresponding to $\lambda = 0.01$.

## 6  Discussion.

We have introduced a general approach to including a preference for diversity into generative probabilistic models. We showed how a determinantal point process can be integrated as a modular component into existing learning algorithms, and discussed its general role as a diversity regularizer. We investigated two settings where diversity can be useful: learning topics from documents, and clustering image patches. Plugging a DPP into latent Dirichlet allocation allows LDA to automatically group stop-words into a few categories, enabling more informative topics in other categories. In both document and image classification tasks, there exists an intermediate regime of diversity (as controlled by the hyperparameter $\lambda$) that leads to consistent improvement in accuracy when compared to standard i.i.d. models. A computational bottleneck can come from inverting the $M \times M$ kernel matrix $\mathbf{K}$, where $M$ is the number of latent distributions. However in many settings such as LDA, $M$ is much smaller than the data size. We expect that there are many other settings where DPP-based diversity can be usefully introduced into a generative probabilistic model: in the emission parameters of HMM and more general time series, and as a mechanism for transfer learning.

# References

[1] David M. Blei, Andrew Y. Ng, and Michael I. Jordan. Latent Dirichlet allocation. *Journal of Machine Learning Research*, 3:993–1022, 2003.

[2] J. F. C. Kingman. *Poisson Processes*. Oxford University Press, Oxford, United Kingdom, 1993.

[3] David J. Strauss. A model for clustering. *Biometrika*, 62(2):467–475, August 1975.

[4] Jesper Møller and Rasmus Plenge Waagepetersen. *Statistical Inference and Simulation for Spatial Point Processes*. Monographs on Statistics and Applied Probability. Chapman and Hall/CRC, Boca Raton, FL, 2004.

[5] J. Ben Hough, Manjunath Krishnapur, Yuval Peres, and Blint Virág. Determinantal processes and independence. *Probability Surveys*, 3:206–229, 2006.

[6] Antonello Scardicchio, Chase E. Zachary, and Salvatore Torquato. Statistical properties of determinantal point processes in high-dimensional Euclidean spaces. *Physical Review E*, 79(4), 2009.

[7] Frédéric Lavancier, Jesper Møller, and Ege Rubak. Statistical aspects of determinantal point processes. `http://arxiv.org/abs/1205.4818`, 2012.

[8] Alex Kulesza and Ben Taskar. Structured determinantal point processes. In *Advanced in Neural Information Processing Systems 23*, 2011.

[9] Alex Kulesza and Ben Taskar. Learning determinantal point processes. In *Proceedings of the 27th Conference on Uncertainty in Artificial Intelligence*, 2011.

[10] Tony Jebara, Risi Kondor, and Andrew Howard. Probability product kernels. *Journal of Machine Learning Research*, 5:819–844, 2004.

[11] Adam Coates Honglak Lee and Andrew Ng. An analysis of single-layer networks in unsupervised feature learning. In *Proceedings of the 14th International Conference on Artificial Intelligence and Statistics*, 2011.

